# Learning Structural Equation Models for fMRI

**Amos J. Storkey**
School of Informatics
University of Edinburgh

**Enrico Simonotto**
Division of Psychiatry
University of Edinburgh

**Heather Whalley**
Division of Psychiatry
University of Edinburgh

**Stephen Lawrie**
Division of Psychiatry
University of Edinburgh

**Lawrence Murray**
School of Informatics
University of Edinburgh

**David McGonigle**
Centre for Functional Imaging Studies
University of Edinburgh

## Abstract

Structural equation models can be seen as an extension of Gaussian belief networks to cyclic graphs, and we show they can be understood generatively as the model for the joint distribution of long term average equilibrium activity of Gaussian dynamic belief networks. Most use of structural equation models in fMRI involves postulating a particular structure and comparing learnt parameters across different groups. In this paper it is argued that there are situations where priors about structure are not firm or exhaustive, and given sufficient data, it is worth investigating learning network structure as part of the approach to connectivity analysis. First we demonstrate structure learning on a toy problem. We then show that for particular fMRI data the simple models usually assumed are not supported. We show that is is possible to learn sensible structural equation models that can provide modelling benefits, but that are not necessarily going to be the same as a true causal model, and suggest the combination of prior models and learning or the use of temporal information from dynamic models may provide more benefits than learning structural equations alone.

## 1 Introduction

Structural equation modelling (SEM) is a technique widely used in the behavioural sciences. It has also appeared as a standard approach for analysis of what has become known as *effective connectivity* in the functional magnetic resonance imaging (fMRI) literature and is still in common use despite the increasing interest in dynamical methods such as dynamic causal models [6]. Simply put, effective connectivity analysis involves looking at the possible causal influences between brain regions given measurements of the activity of those regions. Structural equation models are a Gaussian modelling tool, and are similar to Gaussian belief networks. In fact Gaussian belief networks can be seen as a subset of valid structural equation models. However structural equation models do not have the same acyclicity constraints as belief networks. It should be noted that the graphical form used in this paper is at odds with traditional SEM representations, and consistent with that used for belief networks, as those will be more familiar to the expected audience.

Within the fMRI context, the use of structural equation generally takes the following form. First certain regions of interests (commonly called *seeds*) are chosen according to some understanding of what brain regions might be of interest or of importance. Then neurobiological knowledge is used to propose a connectivity model. This connectivity model states what regions are connected to what other regions, and the direction of the connectivity. This connectivity model is used to define a structural equation model. The parameters of this model are then typically estimated using maximum likelihood methods, and then comparison of connection parameters is made across subject classes.

In this paper we consider what can be done when it is hard to specify connectivity a priori, and ask how much we can achieve by learning network structures from the fMRI data itself. The novel developments of this paper include the examination of various generative representations for structural equation models which allow straightforward comparisons with belief networks and other models such as dynamic causal models. We implement Bayesian Information Criterion approximations to the evidence and use this in a Metropolis-Hastings sampling scheme for learning structural equation models. These models are then applied to toy data, and to fMRI data, which allows the examination of the types of assumptions typically made.

## 1.1 Related Work: Structural Equation Models

Structural equation models and path analysis have a long history. The methods were introduced in the context of genetics in [20], and in econometrics in [7]. They have been used extensively in the social sciences in a variety of ways. Linear Gaussian structural equation models can be split into the case of path analysis [20], where the all the variables are directly measurable and structural equation models with latent variables [1], where latent variable models are allowed. Factor analysis is another special case of this latter. Furthermore structural equation models can also be characterised by the inclusion of exogenous influences.

Structural equation models have been analysed and understood in Bayesian terms before. They form a part of the causal modelling framework of Pearl [11], and have been discussed within that context, as well as a number of others [11, 4, 13, 10]. Approaches to learning structural equation models have not played a significant part in fMRI methods. One approach is described in [3] where they use a genetic algorithm approach for the search. In [21], the authors look at learning Bayesian networks but do not consider cyclic networks. For dynamic causal models (rather than structural equation models) the issue of model comparison was dealt with in [12], but large scale structure learning was not considered.

In fMRI literature, SEMs have generally been used to model 'effective connectivity', or rather modelling the causal relationships between different brain regions. They were first applied to imaging data by [9], and there have been many further applications [2, 5, 14]. The first analysis on data from schizophrenia studies was detailed in [15]. In fact it seems SEMs have been the most widely used model for connectivity analyses in neuroimaging. In all of the studies cited above the underlying structure was presumed known or presumed to be one of a small number of possibilities. There has been some discussion of how best to obtain reasonable structures from neuro-anatomical data, but this approach is currently used only very rarely.

## 2 Why Learn SEMs?

The presumption in much fMRI connectivity analysis is that we can obtain models for activity dependence from neuro-anatomical sources. The problem with this is that it fails to account for the fact that connectivity analysis is usually done with a limited number of regions. It is highly possible that a connection from one region to another is mediated via a third region, which is not included in the SEM model. The strength of that mediation is unknown from neuro-anatomical data and is generally ignored: most connectivity models focus only on direct anatomical connections, with the accompanying implicit assumption that there are no other regions involved in the network under study, or that these regions would contribute only minimally to the model. Furthermore, just because regions are physically connected does not mean there is any actual functional influence in a particular context. Hence it has to be accepted that neuro-anatomically derived connectivity is a first guess at best.

It is not the purpose of this paper to propose that anatomical connectivity be ignored, but instead it asks what happens if we go to the other extreme: can we say something about connectivity from the data? In reality anatomical connectivity models are needed, and can be used to provide good priors for the connections and even for the relative connection strengths. Statistically there are huge equivalences in structural equation models that will not be determined by the data alone.

## 3 Understanding Structural Equation Models

In this section two generative views of structural equation modelling are presented. The idea behind structural equation modelling is that it represents causal dependence between different variables. The fact that cyclic structures are allowed in structural equation models could be seen as an implicit assumption of some underlying dynamic which the structural equation model is an equilibrium rep-

resentation of. Indeed that is commonly how effective connectivity models are interpreted in an fMRI context. Two linear models, both of which produce a structural equation model prior, are presented here. Though these models have the same statistical properties, they have different generative motivations and different non-linear extensions, so they are both potentially instructive.

## 3.1 The Traditional Model

The standard SEM view is that the core SEM structure is a covariance produced by the solution to a set of linear equations $\mathbf{x} = A\mathbf{x} + \boldsymbol{\omega}$ with Gaussian term $\boldsymbol{\omega}$. This does not have any direct generative elucidation, but can instead be thought of as relating to a deterministic dynamical system subject to uncertain fixed input. Suppose we have a dynamical system $\mathbf{x}_{t+1} = A\mathbf{x}_t + \boldsymbol{\omega}$, subject to some input $\boldsymbol{\omega}$, where we presume the system input is unknown and Gaussian distributed. To generate from the model, we sample $\boldsymbol{\omega}$, run the dynamical system to its fixed point, and use that fixed point as a sample of $\mathbf{x}$. This fixed point is given by $\mathbf{x} = (I - A)^{-1}\boldsymbol{\omega}$ which produces the standard SEM covariance structure for $\mathbf{x}$. This requires $A$ to be a contraction map to obtain stable fixed points. All the other aspects of the general form of SEM are either inputs to or measurements from this system.

## 3.2 Average Activity Of A Gaussian Dynamic Bayesian Network

An alternative and potentially appealing view is that the the SEM represents the distribution of the long term activity of the nodes in a Gaussian dynamic Bayesian network (Kalman filter). Suppose we have $\mathbf{x}_t = A\mathbf{x}_{t-1} + \boldsymbol{\omega}_t$, where $\boldsymbol{\omega}_t$ are IID Gaussian variables, and $\mathbf{x}_0, \mathbf{x}_1, \ldots$ is a series of real variables. This defines a Markov chain, and is the evolution equation of a Gaussian dynamic Bayesian network. Suppose we are at the equilibrium distribution of this Markov chain. Then setting $\tilde{\mathbf{x}} = (1/\sqrt{N})\sum_{t=1}^{N}\mathbf{x}_t$ for large $N$, we can use the Kalman filter to see that $(1/\sqrt{N})\sum_{t=1}^{N}\mathbf{x}_t = (1/\sqrt{N})[A(\mathbf{x}_0 - \mathbf{x}_N) + A\sum_{i=1}^{N}\mathbf{x}_t] + (1/\sqrt{N})\sum_{t=1}^{N}\boldsymbol{\omega}_t$. Presuming $A$ is a contraction map, $(1/\sqrt{N})[A(\mathbf{x}_0 - \mathbf{x}_N)]$ becomes negligibly small and so $\tilde{\mathbf{x}} \approx A\tilde{\mathbf{x}} + \boldsymbol{\omega}$ where $\boldsymbol{\omega}$ is distributed identically to $\boldsymbol{\omega}_t$ due to the fact that the variance of a sum of Gaussians is the sum of the variances. The approximation becomes an equality in the large $N$ limit. Again this is the required form for obtaining the covariance of the SEM.

This interpretation says that if we have some latent system running as a Gaussian dynamic Bayesian network, but our measuring equipment is only capable of capturing longer term averages of the network activity then our measurements are distributed according to an SEM. This generative interpretation is appealing in the context of fMRI acquisition. Note in both of these interpretations that it is important that $A$ is a contraction. By formulating the generative framework we see it is important to restrict the form of connectivity model in this way.

# 4 Model Structure

The standard formalism for Structural Equation Models is now outlined. A structural equation model for observational variables $\mathbf{y}$, latent variables $\mathbf{x}$ and sometimes for latent input variables $\boldsymbol{\phi}$ and observations of the input variables $\mathbf{z}$ is given by the following equations

$$\mathbf{x} = (I - A)^{-1}(R\boldsymbol{\phi} + \boldsymbol{\omega}), \mathbf{y} = B\mathbf{x} + \boldsymbol{\sigma} \text{ and } \mathbf{z} = C\boldsymbol{\phi} + \boldsymbol{\delta} \tag{1}$$

where $\boldsymbol{\sigma}$, $\boldsymbol{\omega}$, $\boldsymbol{\phi}$ and $\boldsymbol{\delta}$ are Gaussian, and $A$ is presumed to be zero diagonal.

For for $S = I - A$, the resulting covariance for the observed variables $(\mathbf{y}, \mathbf{z})$ is given by

$$\begin{matrix} BS^{-1}(RK_\phi R^T + K_\omega)[S^{-1}]^T + K_\sigma & BS^{-1}RK_\phi C^T \\ CK_\phi R^T[S^{-1}]^T B^T & CK_\phi C^T + K_\delta \end{matrix} . \tag{2}$$

where $K_\omega$ is the covariance of $\boldsymbol{\omega}$, $K_\sigma$ the covariance of $\boldsymbol{\sigma}$ etc. There are a number of common simplifications to this framework. The first case involves presuming no inputs and a fully visible system. Hence we marginalise the observations of the input variables $\mathbf{z}$, set $K_\delta = \infty$, $C = 0$ , $R = 0$, $B = 1$, $\sigma = 0$. Then the covariance $K_1$ of $\mathbf{y}$ is $K_1 = (I - A)^{-1}K_\omega[(I - A)^{-1}]^T$.

The next simplest case would involve presuming once again that there are no inputs but that in fact the observations are stochastic functions of the latent variables. This involves setting $K_\delta = \infty$, $C = 0$, $R = 0$, $B = 1$. We then have $K_2 = (I - A)^{-1}K_\omega[(I - A)^{-1}]^T + K_\sigma$. If we view the observations as noisy versions of the latent variables then $K_\sigma$ is diagonal. This will be the most general case considered in this paper. Adding any of the remaining components is not particularly demanding as it simply uses a conditional rather than unconditional model.

Suppose we denote by $K$ the covariance corresponding to the required model. For most of this paper we presume $K = K_2$. We then have the following probability for the whole data $Y = \{y_1, y_2, \ldots, y_N\}$.

$$P(Y|K, \bar{\mathbf{y}}) = \prod_j \frac{1}{(2\pi)^{m/2}|K|^{1/2}} \exp\left(-\frac{1}{2}(\mathbf{y}^j - \bar{\mathbf{y}})^T K^{-1}(\mathbf{y}^j - \bar{\mathbf{y}})\right) \tag{3}$$

where the observable model mean is $\bar{\mathbf{y}} = \bar{\mathbf{x}} + \bar{\boldsymbol{\sigma}}$ and the latent mean is $\bar{\mathbf{x}} = (I - A)^{-1}\bar{\boldsymbol{\omega}}$, and where $\boldsymbol{\sigma}$ and $\boldsymbol{\omega}$, along with elements of the matrix $A$ and covariances $K_\omega$ and $K_\sigma$, are parameters.

## 5  Priors, Maximum Posterior and Bayesian Information Criterion

The previous section outlines the basic model of the data given the various parameters. In this section we provide prior distributions for the parameters of the structural equation model. Independent Gaussian priors are put on the parameters:

$$P(A_{ij}|T) = \frac{T^{1/2}}{(2\pi)^{1/2}} \exp\left(-\frac{1}{2}T(A_{ij} - \bar{A}_{ij})^2\right) \tag{4}$$

with regularisation parameter $T$. For the purposes of this paper we take $\bar{A}_{ij} = 0$, presume we have no particular a priori bias towards positive or negative connections and a uniform prior over structures. An independent prior over connections seems reasonable as two separate connections between different brain regions would have no a priori reason to be related. Any relationship is due to functional purpose and is therefore a posteriori. The use of a uniform prior over all structures is an extreme position, which we have taken in this paper to contrast with using only one structure. In reality we would want to use neurobiologically guided priors over structures.

Inverse gamma priors were also specified for $K_\omega$ and $K_\sigma$ originally, along with a prior for the mean $\bar{\omega}$. It was found that these typically had no effects on the experiments and were dropped for simplicity. Hence $K_\omega$ and $K_\sigma$ will be optimised without regularisation, and $\bar{\omega}$ is set to zero. $T$ is chosen by 10 fold cross-validation from a set of 10 possible values.

We can calculate all the relevant derivatives for the SEM straightforwardly, and adapt the parameters to maximize the posterior of the structural equation model. In this paper we use a conjugate gradient approach. By adding a Bayesian Information Criterion term [16], $(-0.5m \log N)$ for m parameters and N data points, to the log posterior at the maximum posterior solution, we can obtain an approximation of the evidence $P(Y|M)$ where $M$ encodes the structural information we are interested in and consists of indicator variables $M_{ij}$ indicating a connection for node $j$ to node $i$. This will enable us to sample from an approximate posterior distribution of structures to find a sample which best represents the data.

## 6  Sampling From SEMs

In order to represent the posterior distribution over network structures, we resort to a sampling approach. Because there are no acyclicity constraints, MCMC proposals are simpler than the comparable situation for belief networks in that no acyclicity checking needs to be done for the proposals. A simple proposal scheme is to randomly generate a swap matrix $M_S$ which is XORed with $M$. We choose highly sparse swap matrices, but to reduce the possibility of transitioning randomly about the larger graphs, without ever considering smaller networks we introduce a bias towards removing connections rather than adding connections in generating the swap matrix. This means the proposal is no longer symmetric, and so a corresponding Hastings factor needs to be included in the acceptance probability, so the result is still a sample from the original posterior.

## 7  Tests On A Toy Problem

We tested the approach on a toy problem with 8 variables. We sampled 800 data points from $\mathbf{y} = (I-A)^{-1}\epsilon + \rho$ for $\epsilon$ Gaussian with unit diagonal covariance, $\rho$ Gaussian with 0.2 diagonal covariance and with $A$ given by

$$\begin{pmatrix}
0 & 0 & 0 & -0.26 & 0 & 0 & -0.03 & 0 \\
0 & 0 & 0.47 & 0 & -0.36 & 0.55 & 0 & 0 \\
0 & 0 & 0 & 0 & 0.34 & 0 & 0 & 0 \\
0 & 0 & -0.36 & 0 & 0 & -0.03 & -0.08 & 0.25 \\
0 & 0 & 0.27 & 0 & 0 & 0 & -0.25 & 0 \\
-0.17 & 0.49 & 0 & 0 & -0.18 & 0 & 0 & 0 \\
0.31 & 0.42 & -0.13 & 0 & 0 & 0 & 0 & -0.22 \\
0.385 & 0 & 0 & 0 & 0.16 & 0.5 & 0 & 0
\end{pmatrix}$$

This connectivity matrix is represented graphically in Figure 11. In modelling this we used $T = 10$. This prior ensures that any $A$ that is not of very low prior probability is a contraction. Also contraction constraints were added to the optimisation. Priors on other parameters were set to be broad. An annealed sampling procedure was used for the first 4000 samples from the Metropolis-Hastings Monte-Carlo procedure. After that a further 4000 samples burn-in was used. The next 4000 samples were used for analysis.

We assess what edges are common in the samples, and what the highest posterior sampled graphs are. Figure 1b provides illustrations of edges which are present in more than 0.15 of the samples. It can be seen that many of the critical edges are there in most samples (indeed some are always there). Those which are missing in both cases tend to be either low magnitude connections or are due to directional confusion.

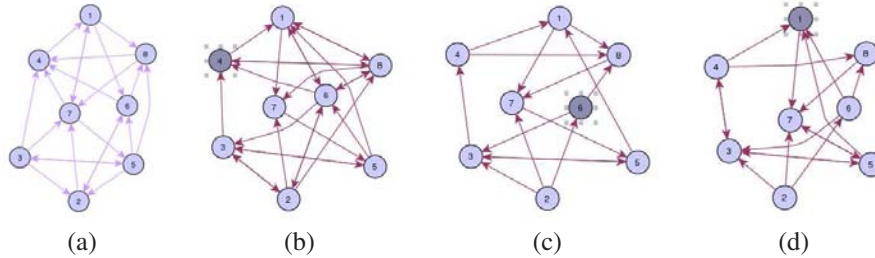

(a)          (b)          (c)          (d)

Figure 1: (a) Graphical structure of the ground truth model for the toy data, and (b) Edges present in more than 0.15 of the cases, (c) the highest posterior structure from the sample (d) a random sample.

The graphs for the maximum posterior sample and a random sample are shown in Figure 1. We can see that again in the maximum posterior sample, there is a misplaced edge (the edge from 5 to 6 is replaced by one from 5 to 1) and a number are missing or have exchanged direction. The samples generally have likelihoods which are very similar to the likelihood for the true model.

We can conclude from this that we can gain some information from learning SEM structures, but as with learning any graphical models there are many symmetries and equivalences, so it is vital not to infer too much from the learnt structures.

## 8   Tests On fMRI Data

The approach of this paper was tested on two different fMRI datasets. The first dataset (D1) was taken from a dataset that had previously used to examine inter-session variance in a single subject [8, 17]. We used the auditory paced finger-tapping task; briefly, a single subject tapped his right index finger, paced by an auditory tone (1.5Hz). Each activation epoch was alternated with a rest epoch, in which the pacing tone was delivered to control for auditory activation. Thirteen blocks were collected per session (seven rest and six active). Each block was 24s/6 scans long, making 78 scans in total for each of 33 sessions. The subject maintained fixation on a cross that was backprojected onto a transparent screen by a LCD video projector as in previous experiments.

The subject was a healthy 23 year old right-handed male. The data were acquired on a Siemens MAGNETOM Vision (Siemens, Erlangen, Germany) at 2T. Each BOLD-EPI volume scan consisted of 48 transverse slices (inplane matrix 64x64; voxel size 3x3x3mm; TE=40ms; TR=4.1s). A T1-weighted high-resolution MRI of the subject (1 x 1 x 1.5mm resolution) was acquired to facilitate anatomical localisation of the functional data.

The data were processed with statistical parametric mapping (SPM) software SPM5 (Wellcome Department of Cognitive Neurology; www.fil.ion.ucl.ac.uk/spm). After removal of the first two volumes to account for T1 saturation effects, cerebral volumes were realigned to correct for both within- and between-session subject motion). The data were filtered with a 128s high-pass filter, and an AR(1)-model was used to account for serial correlation in the data Experimental effects were estimated using session design matrices modeling the hemodynamically convolved time-course of the active movement condition, and 6 subject movement parameters. Note that no spatial smoothing was applied to this dataset, to attempt to preserve single-voxel timeseries.

Seeds were selected from significantly active voxels identified using a random effects analysis in SPM5 (ones-sample t-test across 33 sessions; $p < 0.05$ FWE corrected for multiple comparsions).

For comparison with previous extant work, the most significant voxel in each cluster was chosen as a seed, giving 13 seeds representing 13 separate anatomical regions. When it was obvious that a given cluster encompassed more than one distinct anatomical region, seeds were also selected for other regions covered by the cluster. 2000 data points were used for training, the remaining 574 were reserved as a test set.

The second dataset (D2) was from a long term study of subjects who are at genetically enhanced risk of schizophrenia. Imaging was carried out on 90 subjects at the Brain Imaging Research Centre for Scotland (Edinburgh, Scotland, UK) on a GE 1.5 T Signa scanner. A high resolution structural scan was acquired using a 3D T1-weighted sequence (TI = 600 ms). Functional data was acquired using an EPI sequence. A total of 204 volumes were acquired. The first four volumes of each acquisition were discarded. Preliminary analysis was carried out using SPM99. Data were first realigned to correct for head movement, normalized to the standard EPI template and smoothed.

The resulting data consists of a image-volume series of 200 time points for each of the remaining 90 patients. The voxel time courses were temporally filtered. In order to reduce task related effects, we modelled the task conditions with standard block effects (Hayling), all convolved with canonical hemodynamical response functions, and fitted a general linear model (which also included regressors for the estimated head movement) to the time filtered data; the residuals of this procedure were used as the data for all the work described in this paper. The full data set was split into two halves, a training and a test set. Data from 45 of the subjects was used for training and 45 for testing.

For an effective connectivity analysis, a number of brain regions (seeds) were chosen on the basis of the results of a functional connectivity study [19] and taking regard of areas which may be of particular clinical interest. In total 14 regions were chosen, along with their 14 cross-hemisphere counterparts. Hence we are interested in learning a 28 by 28 connectivity matrix.

## 8.1 Learning SEM Structure

For both datasets a similar procedure to the toy example was followed for learning structure for the fMRI data. The stability of the log posterior along with estimations of cross-correlation against lag were used as heuristics to determine convergence prior to obtaining 10000 sample points.

Assuming a fully visible path analysis (covariance $K_1$) model, where no measurement noise is included is typical in fMRI analysis (e.g. [15] for a Schizophrenia studies), we found that samples from the posterior of this model were in fact so highly connected that displaying them would infeasible. For D2 a connectivity of 350 of the 752 total possible connections was typical. However note that only 376 connections are needed to fully specify a general covariance. Hence we can assume that in this situation the data is not suggesting any particular structure in the data which is reasonably amenable to path analysis.

We can generalise the path analysis model by making the region activities latent variables, and allow the measurement variables to be noisy versions of those regions. In SEM terms this is equivalent to assuming the covariance structure given by $K_2$. A repeat of the whole procedure with this covariance results in much smaller structures. We focus on this approach.

For dataset $D1$, we sample posterior structures given the training data with $T = 100$. There is notable variation in these structures although some key links (eg Left Motor Cortex (L M1) to Left Posterior Parietal Cortex (L PPC) are included in most samples. In addition an a priori connectivity

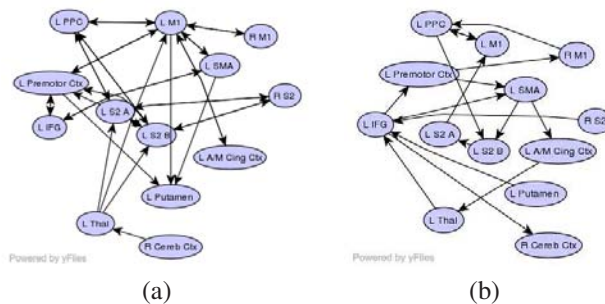

(a)                                        (b)

Figure 2: Structure for (a) the hand specified model (b) the highest posterior sample.

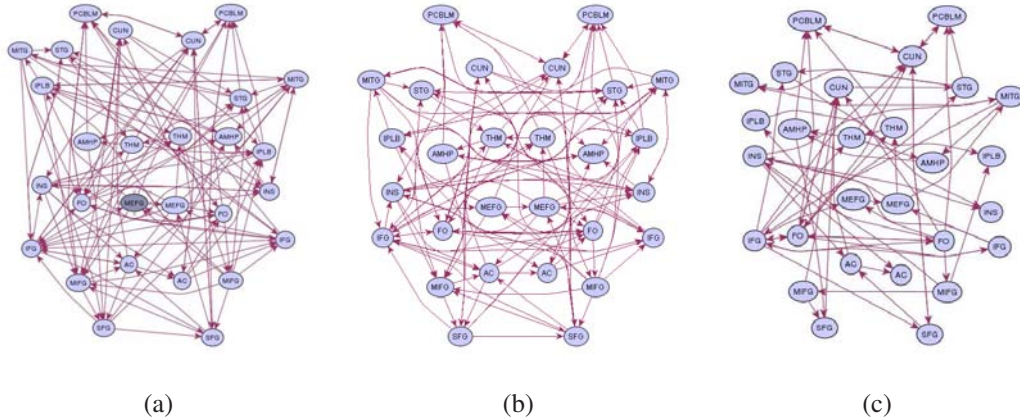

|  (a)  |  (b)  |  (c)  |

Figure 3: Graphical structure for (a) the highest posterior structure from the sample (b) random sample. (c) a sample from the two tier model. The regions are Inferior Frontal Gyrus, Medial Frontal Gyrus, Ant. Cingulate, Frontal Operculum, Superior Frontal Gyrus, Middle Frontal Gyrus, Superior Temporal Gyrus, Middle Temporal Gyrus, Insula, Thalamus, Amygdala Hippocampal Region, Cuneus/Precuneus, Inferior Parietal Lobule and Posterior Cerebellum.

structure is proposed for the regions in the study, taking into account the task involved. This was obtained by using knowledge of neuroanatomical connectivity drawn from studies using tract-tracing in non-human primates. It was produced independent of the connectivity analysis and without knowledge of its results, but taking into account the seed locations and their corresponding activities. Note that this is a simple finger tapping motor task with seeds corresponding to the associated regions. Though not trivial, we would expect the specification to be easier and more accurate here than for more complicated cognitive tasks, due to the high number of papers using this task in functional neuroimaging. Task D1 is also of note due to its focus on repeated scanning in a single individual, thus negating any problems in seed selection that may arise from inter-subject spatial variance.

These two cases described above are specified as different hypothesised models. We denote the hand-specified structure $M_H$ and we select the maximum a posteriori sample $M_L$ (for "Learnt Model") as a potential alternative. The two structures are illustrated in Figure 2. The maximum a posteriori parameters are then estimated for the two models using the same conjugate gradient procedure on the same dataset. These two models are then used predictively on the remaining unseen test data. We compute the predictive log-likelihoods for each model. We find that the best predictive log-likelihoods for each approach are the same (3SF) for both models. They are also the same as the predictive likelihood using the full sample covariance, which given the large data sizes used is well specified. Both these models perform better than other random models with equivalent numbers of connections. In reality learnt models are going to be used in new situations and situations with less data. One test of the appropriateness of a model is to assess its predictive capability when trained on little data. By estimating the model parameters on 100 data points, instead of 2000, we find that the learnt model performs very slightly better than the hand specified model (log odds ratio of 63 on a 574 point test set), and both perform better than the full covariance (log odds of 292). This indicates that both $M_H$ and $M_L$ are providing salient reduced representations which capture useful characteristics of the data.

We also ran tests on D2. Maximum posterior samples and a random sample are illustrated in Figure 3. Note that although these samples appear to still be quite highly connected, they in fact have about 130 connections. Even so this is significantly greater than the idealised connectivity structures typically used in most studies.

One further approach is to assume a fully connected structure, but where the connectivity is in two categories. We have priors on connectivity with the same values of $T_{ij}$ as before for the strong connections and much larger values for the weaker connections. When this is added to the form of the model (where we make the incorrect but practical assumption that the BIC assumption still holds for the stronger connections) we obtain even simpler structures. Following this procedure we find that models of the form of 3c are typical samples from the posterior where only the larger

connections are shown. Again connections such as those between the Cuneus/Precuneus and the Superior Frontal Gyrus, the Thalamic connections, and some of the cross-hemispheric connections are amongst those that would be expected. This approach is related to recent work on the use of sparse priors for effective connectivity [18].

## 9 Future Directions

This work demonstrates that if we learn structural equation models from data, we find there is little evidence for the simple forms of path analysis model which is in common use in the fMRI literature. We suggest that learning connectivity can be a reasonable complement to current procedures where prior specification is hard. Learning on its own does discover useful parameterised representations, but these parameterisations are not the same as reasonable prior specifications. This is unsurprising due to the statistical equivalence of many SEM structures. It should be expected that combining learnt structures with prior anatomical models will help in the specification of more accurate connectivity assumptions, as it will reduce the number of equivalence and focus on more reasonable structural forms. Furthermore future comparisons can be made using a sample of reasonable models instead of a single a priori chosen model. We would also expect that the major gains in learning models with come from the focus on dynamical networks which do not suffer from specificity problems. Even if the level of temporal information is small, any temporal information provides handles for inferring causality that are unavailable with static equilibrium models.

## References

[1] K. A. Bollen. *Structural Equations with Latent Variables*. John Wiley and Sons, 1989.

[2] C. Buchel, J.T. Coull, and K.J. Friston. The preedictive value of changes in effective connectivity for human learning. *Science*, 283:1528–1541, 1999.

[3] E. Bullmore, B. Howitz, G. Honey, M. Brammer, S. Williams, and T. Sharma. How good is good enough in path analysis of fMRI data? *Neuroimage*, 11:289–301, 2000.

[4] D. Dash. Restructing dynamic causal systems in equilibrium. In *Proc. Uncertainty in AI 2005*, 2005.

[5] K.J. Friston and C. Buchel. Attentional modulation of effective connectivity from V2 to V5/MT in humans. *Proceedings of the National Academy of Sciecnes*, 97:7591–7596, 2000.

[6] K.J. Friston, L. Harrison, and W.D. Penny. Dynamic causal modelling. *NeuroImage*, 19:1273–1302, 2003.

[7] T. Haavelmo. The statistical implications of a system of simultaneous equations. *Econometrica*, 11:1–12, 1943.

[8] D. McGonigle, A. Howseman, B. Athwal, K.J. Friston, R. Frackowiak, and A. Holmes. Variability in fmri: An examination of intersession differences. *Neuroimage*, 11:708–734, 2000.

[9] A. R. McIntosh and F. Gozales-Lima. Structural equation modelling and its application to network analysis in functional brain imaging. *Human Brain Mapping*, 2:2–22, 1994.

[10] C. Glymour P. Spirtes and R. Scheines. *Causation, Prediction and Search*. MIT Press, 2 edition, 2001.

[11] J. Pearl. *Causality*. Cambridge University Press, 2000.

[12] W.D. Penny, K.E. Stephan, A. Mechelli, and K.J. Friston. Comparing dynamic causal models. *Neuroimage*, 22:1157–1172, 2004.

[13] T. Richardson. A discovery algorithm for directed cyclic graphs. In *Proceedings of the 12th Conference on Uncertainty in Artificial Intelligence*, 1996.

[14] J. Rowe, K.E. Stephan, K. Friston, R. Frackowiak, A. Lees, and R. Passingham. Attention to action in Parkinsons disease. *Brain*, 125:276–289, 2002.

[15] R. Schlosser, T. Gesierich, B. Kauffman, G. Vucurevic, S. Hunsche, J. Gawehn, and P. Stoeter. Altered effective connectivity during working memory performance in schizophrenia: a study with fMRI and structural equation modeling. *Neuroimage*, 19:751–763, 2003.

[16] G. Schwarz. Estimating the dimension of a model. *Annals of Statistics*, 6:461–464, 1978.

[17] S.M.Smith, C.F. Beckmann, N. Ramnani, M.W. Woolrich, P.R. Bannister, M. Jenkinson, P.M. Matthews, and D. McGonigle. Variability in fMRI: A re-examination of intersession differences. *Human Brain Mapping*, 24:248–257, 2005.

[18] P.A. Valdes Sosa, J.M. Sanchez-Bornot, A. Lage-Castellanos, M. Vega-Hernandez, J. Bosch Bayard, L. Melie-Garcia, and E. Canales-Rodriguez. Estimating brain functional connectiivty with sparse multivariate autoregression. *Philosophical Transactions of the Royal Society onf London B Biological Sciences*, 360:969–981, 2005.

[19] H.C. Whalley, E. Simonotto, I. Marshall, D.G.C Owens, N.H. Goddard, E.C. Johnstone, and S.M. Lawrie. Functional disconnectivity in subjects at high genetic risk of schizophrenia. *Brain*, 128:2097–2108, 2005.

[20] S. Wright. Correlation and causation. *Journal of Agricultural Research*, 20:557–585, 1921.

[21] X. Zheng and J. C. Rajapakse. Learning functional structure from fMR images. *Neuroimage*, 31:1601–1613, 2006.
